# Unsupervised Discrimination of Clustered Data via Optimization of Binary Information Gain

**Nicol N. Schraudolph**
Computer Science & Engr. Dept.
University of California, San Diego
La Jolla, CA 92093–0114
*nici@cs.ucsd.edu*

**Terrence J. Sejnowski**
Computational Neurobiology Laboratory
The Salk Institute for Biological Studies
San Diego, CA 92186-5800
*tsejnowski@ucsd.edu*

## Abstract

We present the information-theoretic derivation of a learning algorithm that clusters unlabelled data with linear discriminants. In contrast to methods that try to preserve information about the input patterns, we maximize the information gained from observing the output of robust binary discriminators implemented with sigmoid nodes. We derive a local weight adaptation rule via gradient ascent in this objective, demonstrate its dynamics on some simple data sets, relate our approach to previous work and suggest directions in which it may be extended.

## 1  INTRODUCTION

Unsupervised learning algorithms may perform useful preprocessing functions by pre-serving some aspects of their input while discarding others. This can be quantified as maximization of the information the network's output carries about those aspects of the input that are deemed important.

(Linsker, 1988) suggests maximal preservation of information about all aspects of the input. This *Infomax* principle provides for optimal reconstruction of the input in the face of noise and resource limitations. The *I-max* algorithm (Becker and Hinton, 1992), by contrast, focusses on coherent aspects of the input, which are extracted by maximizing the mutual information between networks looking at different patches of input.

Our work aims at recoding *clustered* data with adaptive discriminants that selectively emphasize gaps between clusters while collapsing patterns within a cluster onto near-

identical output representations. We achieve this by maximizing *information gain* — the information gained through observation of the network's outputs under a probabilistic interpretation.

## 2  STRATEGY

Consider a node that performs a weighted summation on its inputs $\vec{x}$ and squashes the resulting net input $y$ through a sigmoid function $f$:

$$z = f(y), \text{ where } f(y) = \frac{1}{1 + e^{-y}} \text{ and } y = \vec{w} \cdot \vec{x}. \tag{1}$$

Such a sigmoid node can be regarded as a "soft" discriminant: with a large enough weight vector, the output will essentially be binary, but smaller weights allow for the expression of varying degrees of confidence in the discrimination.

To make this notion more precise, consider $y$ a random variable with bimodal distribution, namely an even mixture of two Gaussian distributions. Then if their means equal $\pm$ half their variance, $z$ is the posterior probability for discriminating between the two source distributions (Anderson, 1972).

This probabilitstic interpretation of $z$ can be used to design a learning algorithm that seeks such bimodal projections of the input data. In particular, we search for highly informative discriminants by maximizing the information gained about the binary discrimination through observation of $z$. This *binary information gain* is given by

$$\Delta H(z) = H(\hat{z}) - H(z), \tag{2}$$

where $H(z)$ is the entropy of $z$ under the above interpretation, and $\hat{z}$ is an estimate of $z$ based on prior knowledge.

## 3  RESULTS

### 3.1  THE ALGORITHM

In the Appendix, we present the derivation of a learning algorithm that maximizes binary information gain by gradient ascent. The resulting weight update rule is

$$\Delta \vec{w} \propto f'(y) \, \vec{x} \, (y - \hat{y}), \tag{3}$$

where $\hat{y}$, the estimated net input, must meet certain conditions[1] (see Appendix). The weight change dictated by (3) is thus proportional to the product of three factors:

- the derivative of the sigmoid squashing function,
- the presynaptic input $\vec{x}$, and
- the difference between actual and anticipated net input.

$$\Delta y = \tfrac{\partial}{\partial y} \, \Delta H(z)$$

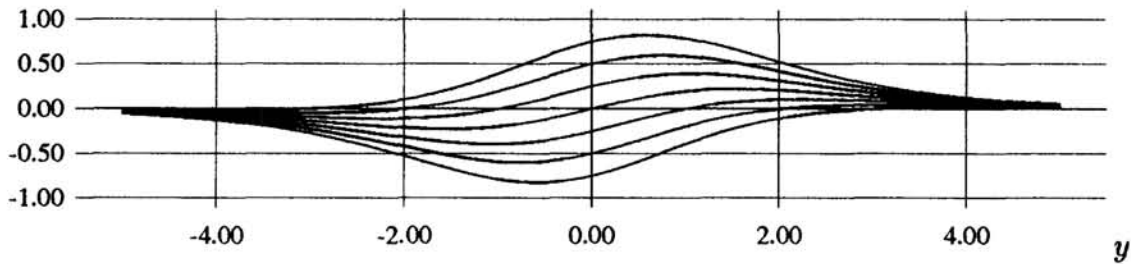

Figure 1: Phase plot of $\Delta y$ against net input $y$ for $\hat{y} = \{-3, -2, \ldots 3\}$. See text for details.

## 3.2 SINGLE NODE DYNAMICS

For a single, isolated node, we use $\langle y \rangle$, the average net input over a batch of input patterns, as estimator for $y$. The behavior of our algorithm in this setting is best understood from a *phase plot* as shown in Figure 1, where the change in net input resulting from a weight change according to (3) is graphed against the net input that causes this weight change.

Curves are plotted for seven different values of $\hat{y}$. The central curve ($\hat{y} = 0$) is identical to that of the straightforward Hebb rule for sigmoid nodes: both positive and negative net inputs are equally amplified until they reach saturation. For non-zero values of $\hat{y}$, however, the curves become asymmetric: positive $\hat{y}$ favor negative changes $\Delta y$ and vice versa. For $\hat{y} = \langle y \rangle$, it is easy to see that this will have the effect of centering net inputs around zero.

The node will therefore converge to a state where its output is one for half of the input patterns, and zero for the other half. Note that this can be achieved by *any* sufficiently large weight vector, regardless of its direction! However, since simple gradient ascent is both greedy and local in weight space, starting it from small random initial weights is equivalent to a bias towards discriminations that can be made confidently with smaller weight vectors.

To illustrate this effect, we have tested a single node running our algorithm on a set of vowel formant frequency data due to (Peterson and Barney, 1952). The most prominent feature of this data is a central gap that separates front from back vowels; however, this feature is near-orthogonal to the principal component of the data and thus escapes detection by standard Hebbian learning rules.

Figure 2 shows the initial, intermediate and final phase of this experiment, using a visualization technique suggested by (Munro, 1992). Each plot shows the pre-image of zero net input superimposed on a scatter plot of the data set in input space. The two flanking lines delineate the "active region" where the sigmoid is not saturated, and thus provide an indication of weight vector size.

As demonstrated in this figure, our algorithm is capable of proceeding smoothly from a small initial weight vector that responds in principal component direction to a solution which uses a large weight vector in near-orthogonal direction to successfully discriminate between the two data clusters.

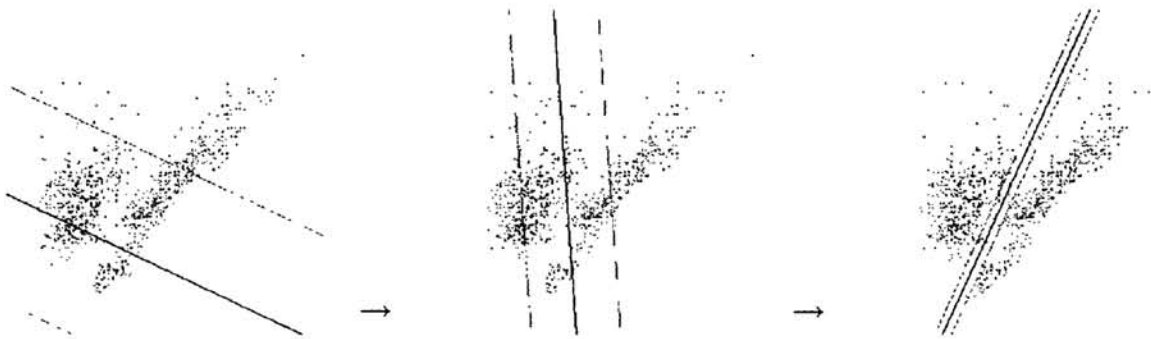

Figure 2: Single node discovers distinction between front and back vowels in unlabelled data set of 1514 multi-speaker vowel utterances (Peterson and Barney, 1952). Superimposed on a scatter plot of the data are the pre-images of $y = 0$ (solid center line) and $y = \pm 1.31696$ (flanking lines) in input space. Discovered feature is far from principal component direction.

## 3.3   EXTENSION TO A LAYER OF NODES

A learning algorithm for a single sigmoid node has of course only limited utility. When extending it to a layer of such nodes, some form of lateral interaction is needed to ensure that each node makes a different binary discrimination. The common technique of introducing lateral competition for activity or weight changes would achieve this only at the cost of severely distorting the behavior of our algorithm.

Fortunately our framework is flexible enough to accommodate lateral differentiation in a less intrusive manner: by picking an estimator that uses the activity of every other node in the layer to make its prediction, we force each node to maximize its information gain with respect to the entire layer. To demonstrate this technique we use the linear second-order estimator

$$\hat{y}_i = \langle y_i \rangle + \sum_{j \neq i} (y_j - \langle y_j \rangle)\, \varrho_{ij} \tag{4}$$

to predict the net input $y_i$ of the $i^{th}$ node in the layer, where the $\langle \cdot \rangle$ operator denotes averaging over a batch of input patterns, and $\varrho_{ij}$ is the empirical correlation coefficient

$$\varrho_{ij} = \frac{\langle (y_i - \langle y_i \rangle)(y_j - \langle y_j \rangle) \rangle}{\sqrt{\langle (y_i - \langle y_i \rangle)^2 \rangle \langle (y_j - \langle y_j \rangle)^2 \rangle}}. \tag{5}$$

Figure 3 shows a layer of three such nodes adapting to a mixture of three Gaussian distributions, with each node initially picking a different Gaussian to separate from the other two. After some time, all three discriminants rotate in concert so as to further maximize information gain by splitting the input data evenly. Note that throughout this process, the nodes always remain well-differentiated from each other.

For most initial conditions, however, the course of this experiment is that depicted in Figure 4: two nodes discover a more efficient way to discriminate between the three input clusters, to the detriment of the third. The latecomer repeatedly tries to settle into one of the gaps in the data, but this would result in a high degree of predictability. Thus the node with the shortest weight vector and hence most volatile discriminant is weakened further, its weight vector all but eliminated in an effective demonstration of Occam's razor.

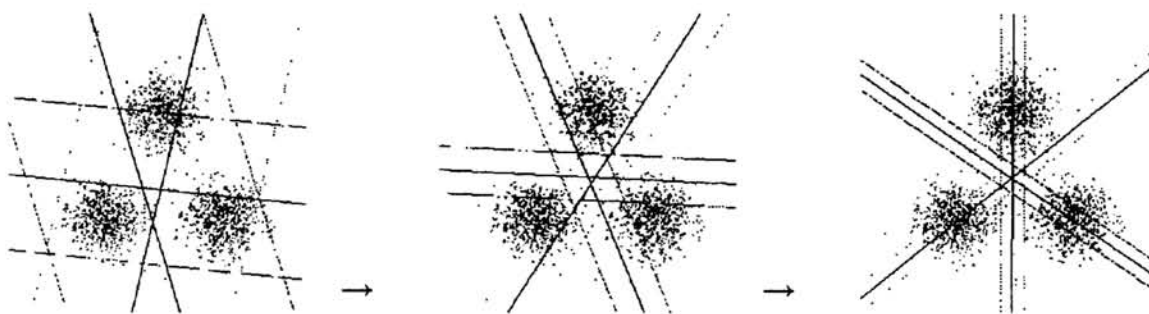

Figure 3: Layer of three nodes adapts to a mixture of three Gaussian distributions. In the final state, each node splits the input data evenly.

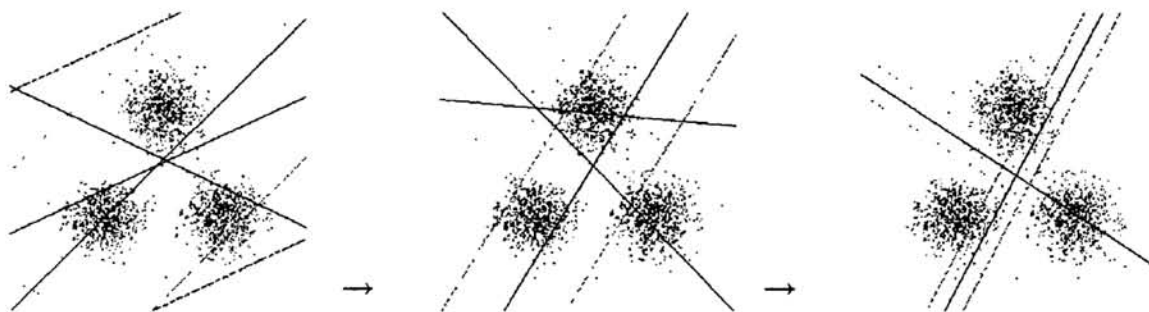

Figure 4: Most initial conditions, however, lead to a minimal solution involving only two nodes. The weakest node is "crowded out" by Occam's razor, its weight vector reduced to near-zero length.

## 4  DISCUSSION

### 4.1  RELATED WORK

By maximizing the difference of actual from anticipated response, our algorithm makes binary discriminations that are highly informative with respect to clusters in the input. The weight change in proportion to a *difference* in activity is reminiscent of the *covariance rule* (Sejnowski, 1977) but generalizes it in two important respects:

- it explicitly incorporates a sigmoid nonlinearity, and
- $\hat{y}$ need not necessarily be the average net input.

Both of these are critical improvements: the first allows the node to respond only to inputs in its non-saturated region, and hence to learn local features in projections other than along the principal component direction. The second provides a convenient mechanism for extending the algorithm by incorporating additional information in the estimator.

We share the goal of seeking highly informative, bimodal projections of the input with the *Bienenstock-Cooper-Munro* (BCM) algorithm (Bienenstock et al., 1982; Intrator, 1992). A critical difference, however, is that BCM uses a complex, asymmetric nonlinearity that increases the *selectivity* of nodes and hence produces a localized, 1-of-n recoding of the input, whereas our algorithm makes symmetric, robust and independent binary discriminations.

## 4.2  FUTURE DIRECTIONS

Since the learning algorithm described here has demonstrated flexibility and efficiency in our initial experiments, we plan to scale it up to address high-dimensional, real-world problems. The algorithm itself is likely to be further extended and improved as its applications grow more demanding.

For instance, although the size of the weight vector represents commitment to a discriminant in our framework, it is not explicitly controlled. The dynamics of weight adaptation happen to implement a reasonable bias in this case, but further refinements may be possible. Other priors implicit in our approach — such as the preference for splitting the data evenly — could be similarly relaxed or modified.

Another attractive generalization of this learning rule would be to implement nonlinear discriminants by backpropagating weight derivatives through hidden units. The dynamic stability of our algorithm is a significant asset for its expansion into an efficient unsupervised multi-layer network.

In such a network, linear estimators are no longer sufficient to fully remove redundancy between nodes. In his closely related *predictability minimization* architecture, (Schmidhuber, 1992) uses backpropagation networks as nonlinear estimators for this purpose with some success.

Since the notion of estimator in our framework is completely general, it may combine evidence from multiple, disparate sources. Thus a network running our algorithm can be trained to complement a heterogeneous mix of pattern recognition methods by maximizing information gain relative to an estimator that utilizes all such available sources of information. This flexibility should greatly aid the integration of binary information gain optimization into existing techniques.

## APPENDIX: MATHEMATICAL DERIVATION

We derive a straightforward batch learning algorithm that performs gradient ascent in the binary information gain objective. On-line approximations may be obtained by using exponential traces in place of the batch averages denoted by the $\langle \cdot \rangle$ operator.

### CONDITIONS ON THE ESTIMATOR

To eliminate the derivative term from (11d) below we require that the estimator $\hat{z}$ be

- unbiased: $\langle \hat{z} \rangle = \langle z \rangle$, and
- *honest*: $\frac{\partial}{\partial z} \hat{z} = \frac{\partial}{\partial z} \langle \hat{z} \rangle$ .

The *honesty* condition ensures that the estimator has access to the estimated variable only on the slow timescale of batch averaging, thus eliminating trivial "solutions" such as $\hat{z} = z$.

For an unbiased and *honest* estimator,

$$\frac{\partial \hat{z}}{\partial z} = \frac{\partial}{\partial z} \langle \hat{z} \rangle = \frac{\partial}{\partial z} \langle z \rangle = \left\langle \frac{\partial z}{\partial z} \right\rangle = 1 \,. \tag{6}$$

## BINARY ENTROPY AND ITS DERIVATIVE

The entropy of a binary random variable $X$ as a function of $z = Pr(X = 1)$ is given by

$$H(z) = - z \log z - (1 - z) \log(1 - z);$$ (7)

its derivative with respect to $z$ is

$$\frac{\partial}{\partial z} H(z) = \log(1 - z) - \log z.$$ (8)

Since $z$ in our case is produced by the sigmoid function $f$ given in (1), this conveniently simplifies to

$$\frac{\partial}{\partial z} H(z) = -y.$$ (9)

## GRADIENT ASCENT IN INFORMATION GAIN

The information $\Delta H$ gained from observing the output $z$ of the discriminator is

$$\Delta H(z) = H(\hat{z}) - H(z),$$ (10)

where $\hat{z}$ is an estimate of $z$ based on prior knowledge. We maximize $\Delta H(z)$ by batched gradient ascent in weight space:

$$\Delta \vec{w} \; \propto \; \left\langle \frac{\partial}{\partial \vec{w}} \Delta H(z) \right\rangle$$ (11a)

$$= \; \left\langle \frac{\partial z}{\partial \vec{w}} \cdot \frac{\partial}{\partial z} [H(\hat{z}) - H(z)] \right\rangle$$ (11b)

$$= \; \left\langle z(1-z) \frac{\partial y}{\partial \vec{w}} \left[ \frac{\partial \hat{z}}{\partial z} \cdot \frac{\partial}{\partial \hat{z}} H(\hat{z}) - \frac{\partial}{\partial z} H(z) \right] \right\rangle$$ (11c)

$$= \; \left\langle z(1-z) \, \vec{x} \left( y - \frac{\partial \hat{z}}{\partial z} \cdot \hat{y} \right) \right\rangle,$$ (11d)

where estimation of the node's output $z$ has been replaced by that of its net input $y$. Substitution of (6) into (11d) yields the binary information gain optimization rule

$$\Delta \vec{w} \propto \langle z(1-z) \, \vec{x} \, (y - \hat{y}) \rangle.$$ (12)

∎

### Acknowledgements

We would like to thank Steve Nowlan, Peter Dayan and Rich Zemel for stimulating and helpful discussions. This work was supported by the Office of Naval Research and the McDonnell-Pew Center for Cognitive Neuroscience at San Diego.

## Footnotes

[1] In what follows, we have successfully used estimators that merely approximate these conditions.

## References

Anderson, J. (1972). Logistic discrimination. *Biometrika*, 59:19–35.

Anderson, J. and Rosenfeld, E., editors (1988). *Neurocomputing: Foundations of Research.* MIT Press, Cambridge.

Becker, S. and Hinton, G. E. (1992). A self-organizing neural network that discovers surfaces in random-dot stereograms. *Nature*, 355:161–163.

Bienenstock, E., Cooper, L., and Munro, P. (1982). Theory for the development of neuron selectivity: Orientation specificity and binocular interaction in visual cortex. *Journal of Neuroscience*, 2. Reprinted in (Anderson and Rosenfeld, 1988).

Intrator, N. (1992). Feature extraction using an unsupervised neural network. *Neural Computation*, 4:98–107.

Linsker, R. (1988). Self-organization in a perceptual network. *Computer*, pages 105–117.

Munro, P. W. (1992). Visualizations of 2-d hidden unit space. In *International Joint Conference on Neural Networks*, volume 3, pages 468–473, Baltimore 1992. IEEE.

Peterson, G. E. and Barney, H. L. (1952). Control methods used in a study of the vowels. *Journal of the Acoustical Society of America*, 24:175–184.

Schmidhuber, J. (1992). Learning factorial codes by predictability minimization. *Neural Computation*, 4:863–879.

Sejnowski, T. J. (1977). Storing covariance with nonlinearly interacting neurons. *Journal of Mathematical Biology*, 4:303–321.
